# Optimal Stopping and Effective Machine Complexity in Learning

**Changfeng Wang**
Department of Systems Sci. and Eng.
University of Pennsylvania
Philadelphia, PA, U.S.A. 19104

**Santosh S. Venkatesh**
Department of Electrical Engineering
University of Pennsylvania
Philadelphia, PA, U.S.A. 19104

**J. Stephen Judd**
Siemens Corporate Research
755 College Rd. East,
Princeton, NJ, U.S.A. 08540

## Abstract

We study the problem of when to stop learning a class of feedforward networks – networks with linear outputs neuron and fixed input weights – when they are trained with a gradient descent algorithm on a finite number of examples. Under general regularity conditions, it is shown that there are in general three distinct phases in the generalization performance in the learning process, and in particular, the network has better generalization performance when learning is stopped at a certain time before the global minimum of the empirical error is reached. A notion of *effective size* of a machine is defined and used to explain the trade-off between the complexity of the machine and the training error in the learning process.

The study leads naturally to a network size selection criterion, which turns out to be a generalization of Akaike's Information Criterion for the learning process. It is shown that stopping learning before the global minimum of the empirical error has the effect of network size selection.

## 1 INTRODUCTION

The primary goal of learning in neural nets is to find a network that gives valid generalization. In achieving this goal, a central issue is the trade-off between the training error and network complexity. This usually reduces to a problem of network size selection, which has drawn much research effort in recent years. Various principles, theories, and intuitions, including Occam's razor, statistical model selection criteria such as Akaike's Information Criterion (AIC) [1] and many others [5, 1, 10, 3, 11] all quantitatively support the following PAC prescription: between two machines which have the same empirical error, the machine with smaller VC-dimension generalizes better. However, it is noted that these methods or criteria do not necessarily lead to optimal (or nearly optimal) generalization performance. Furthermore, all of these methods are valid only at the global minimum of the empirical error function (e.g, the likelihood function for AIC), and it is not clear by these methods how the generalization error is effected by network complexity or, more generally, how a network generalizes during the learning process. This paper addresses these issues.

Recently, it has often been observed that when a network is trained by a gradient descent algorithm, there exists a critical region in the training epochs where the trained network generalizes best, and after that region the generalization error will increase (frequently called over-training). Our numerical experiments with gradient-type algorithms in training feedforward networks also indicate that in this critical region, as long as the network is large enough to learn the examples, the size of the network plays little role in the (best) generalization performance of the network. Does this mean we must revise Occam's principle? How should one define the complexity of a network and go about tuning it to optimize generalization performance? When should one stop learning? Although relevant learning processes were treated by numerous authors [2, 6, 7, 4], the formal theoretical studies of these problems are abeyant.

Under rather general regularity conditions (Section 1), we give in Section 2 a theorem which relates the generalization error at each epoch of learning to that at the global minimum of the training error. Its consequence is that for any linear machine whose VC-dimension is finite but large enough to learn the target concept, the number of iterations needed for the best generalization to occur is at the order of the logarithm of the sample size, rather than at the global minimum of the training error; it also provides bounds on the improvement expected. Section 3 deals with the relation between the size of the machine and generalization error by appealing to the concept of effective size. Section 4 concerns the application of these results to the problem of network size selection, where the AIC is generalized to cover the time evolution of the learning process. Finally, we conclude the paper with comments on practical implementation and further research in this direction.

## 2   THE LEARNING MACHINE

The machine we consider accepts input vectors $X$ from an arbitrary input space and produces scalar outputs

$$y = \sum_{i=1}^{d} \psi_i(X)\alpha^*_i + \xi = \psi(X)'\alpha^* + \xi. \tag{1}$$

Here, $\alpha^* = (\alpha^*_1, \ldots, \alpha^*_d)'$ is a fixed vector of real weights, for each $i$, $\psi_i(X)$ is a fixed real function of the inputs, with $\psi(X) = (\psi_1(X), \ldots, \psi_d(X))'$ the corresponding vector of functions, and $\xi$ is a random noise term. The machine (1) can be thought of as a feedforward neural network with a fixed front end and variable weights at the output. In particular, the functions $\psi_i$ can represent fixed polynomials (higher-order or sigma-pi neural networks), radial basis functions with fixed centers, a fixed hidden-layer of sigmoidal neurons, or simply a linear map. In this context, N. J. Nilsson [8] has called similar structures $\Phi$-machines.

We consider the problem of learning from examples a relationship between a random variable $Y$ and an n-dimensional random vector $X$. We assume that this function is given by (1) for some fixed integer $d$, the random vector $X$ and random variable $\xi$ are defined on the same probability space, that $\mathbf{E}[\xi|X] = 0$, and $\sigma^2(X) = \mathrm{Var}(\xi|X) = constant < \infty$ almost surely. The smallest eigenvalue of the matrix $\psi(x)\psi(x)$ is assumed to be bounded from below by the inverse of some square integrable function.

Note that it can be shown that the VC-dimension of the class of $\Phi$-machines with $d$ neurons is $d$ under the last assumption. The learning-theoretic properties of the system will be determined largely by the eigen structure of $\Phi$. Accordingly, let $\lambda_1 \geq \lambda_2 \geq \cdots \geq \lambda_d$ denote the eigenvalues of $\Phi$.

The goal of the learning is to find the true concept $\alpha$ given independently drawn examples $(X, y)$ from (1). Given any hypothesis (vector) $w = (w_1, \ldots, w_d)'$ for consideration as an approximation to the true concept $\alpha$, the performance measure we use is the mean-square prediction (or ensemble) error

$$\mathcal{E}(w) = \mathbf{E}\left(y - \psi(X)'w\right)^2. \tag{2}$$

Note that the true concept $\alpha^*$ is the mean-square solution

$$\alpha^* = \arg\min_w \mathcal{E}(w) = \Phi^{-1}\mathbf{E}\left(\psi(X)y\right), \tag{3}$$

and the minimum prediction error is given by $\mathcal{E}(\alpha) = \min_w \mathcal{E}(w) = \sigma^2$.

Let $n$ be the number of samples of $(X, Y)$. We assume that an independent, identically distributed sample $(X^{(1)}, y^{(1)}), \ldots, (X^{(n)}, y^{(n)})$, generated according to the joint distribution of $(X, Y)$ induced by (1), is provided to the learner. To simplify notation, define the matrix $\Psi \equiv \left[ \psi(X^{(1)}) \cdots \psi(X^{(n)}) \right]$ and the corresponding vector of outputs $\mathbf{y} = \left( y^{(1)}, \ldots, y^{(n)} \right)'$. In analogy with (2) define the empirical error on the sample by

$$\hat{\mathcal{E}}(w) \equiv \frac{1}{2n} \sum_{j=1}^{t} \left( y^{(j)} - \psi(X^{(j)})'w \right)^2 = \frac{1}{2n} \|\mathbf{y} - \Psi'w\|^2,$$

Let $\hat{\alpha}$ denote the hypothesis vector for which the empirical error on the sample is minimized: $\nabla_w \hat{\mathcal{E}}(\hat{\alpha}) = 0$. Analogously with (3) we can then show that

$$\hat{\alpha} = (\Psi\Psi')^{-1}\Psi\mathbf{y} = \hat{\Phi}^{-1}\left(\frac{1}{n}\Psi\mathbf{y}\right), \tag{4}$$

where $\hat{\Phi} = \frac{1}{n}\Psi\Psi'$ is the empirical covariance matrix, which is almost surely nonsingular for large $n$. The terms in (4) are the empirical counterparts of the ensemble averages in (3).

The gradient descent algorithm is given by:

$$\alpha_{t+1} = \alpha_t - \epsilon\nabla_\alpha \mathcal{E}_n(\alpha_t), \tag{5}$$

where $\alpha = (\alpha_1, \alpha_2, \ldots, \alpha_3)'$, $t$ is the number of iterations, and $\epsilon$ is the rate of learning. From this we can get

$$\alpha_t = (I - \Delta(t))\hat{\alpha} + \Delta(t)\alpha_0, \tag{6}$$

where $\Delta(t) = (I - \epsilon\hat{\Phi})^t$, and $\alpha_0$ is the initial weight vector.

The limit of $\alpha_t$ is $\hat{\alpha}$ when $t$ goes to infinity, provided $\hat{\Phi}$ is positive definite and the learning rate $\epsilon$ is small enough (i.e., smaller than the smallest eigenvalue of $\hat{\Phi}$). This implies that the gradient descent algorithm converges to the least squares solution, starting from any point in $\mathcal{R}^n$.

## 3  GENERALIZATION DYNAMICS AND STOPPING TIME

### 3.1  MAIN THEOREM OF GENERALIZATION DYNAMICS

Even if the true concept (i.e., the precise relation between $Y$ and $X$ in the current problem) is in the class of models we consider, it is usually hopeless to find it using only a finite number of examples, except in some trivial cases. Our goal is hence less ambitious; we seek to find the best approximation of the true concept, the approach entailing a minimization of the training or empirical error, and then taking the global minimum of the empirical error $\hat{\alpha}$ as the approximation. As we have seen the procedure is unbiased and consistent. Does this then imply that training should always be carried out to the limit? Surprisingly, the answer is no. This assertion follows from the next theorem.

**Theorem 3.1** *Let $M_n > 0$ be an arbitrary real constant (possibly depending on $n$), and suppose assumptions A1 to A3 are satisfied; then the generalization dynamics in the training process are governed by the following equation:*

$$\mathcal{E}(\alpha_t) = \mathcal{E}(\alpha_\infty) + \sum_{i=1}^{d} \left[ \lambda_i\delta_i^2(1 - \epsilon\lambda_i)^{2t} - \frac{2\sigma^2}{n}(1 - \epsilon\lambda_i)^t[1 - \frac{1}{2}(1 - \epsilon\lambda_i)^t] \right] + O\left(\frac{M_n^2}{n} + n^{-\frac{3}{2}}\right)$$

*uniformly for all initial weight vectors, $\alpha_0$ in the $d$-dimensional ball $\{\alpha^* + \delta : \|\delta\| \leq M_n, \delta \in \mathcal{R}^d\}$, and for all $t > 0$.* □

## 3.2   THREE PHASES IN GENERALIZATION

By Theorem 3.1, the mean generalization error at each epoch of the training process is characterized by the following function:

$$\phi(t) \equiv \sum_{i=1}^{d} \left[ \lambda_i \delta_i^2 (1 - \epsilon\lambda_i)^{2t} - \frac{2\sigma^2}{n}(1 - \epsilon\lambda_i)^t [1 - \frac{1}{2}(1 - \epsilon\lambda_i)^t] \right].$$

The analysis of the evolution of generalization with training is facilitated by treating $\phi(\cdot)$ as a function of a *continuous* time parameter $t$. We will show that there are three distinct phases in generalization dynamics. These results are given in the following in form by several corollaries of Theorem 3.1.

Without loss of generality, we assume the initial weight vector is picked up in a region with $\|\delta\| \leq M_n = O(n^0)$, and in particular, $|\delta| = O(n^0)$. Let $r_t = \frac{\ln(1/[1-\epsilon\lambda_d])}{\ln n}t$, then for all $0 \leq t < t_1 \equiv \frac{\ln n}{2\ln(1/[1-\epsilon\lambda_d])}$, we have $0 \leq r_t < \frac{1}{2}$, and thus

$$\sum_{i=1}^{d} \lambda_i \delta_i^2 (1 - \epsilon\lambda_d)^{2t} = O(\frac{1}{n^{2r_t}}) >> O(\frac{1}{n^{1+r_t}}) = \frac{2\sigma^2}{n}\sum_{i=1}^{d}(1 - \epsilon\lambda_i)^t.$$

The quantity $\delta_i^2(1 - \epsilon\lambda_i)^{2t}$ in the first term of the above inequalities is related to the elimination of initial error, and can be defined as the approximation error (or fitting error); the last term is related to the effective complexity of the network at $t$ (in fact, an order $O(\frac{1}{n})$ shift of the complexity error). The definition and observations here will be discussed in more detail in the next section.

We call the learning process during the time interval $0 \leq t \leq t_1$ the first phase of learning. Since in this interval $\phi(t) = O(n^{-2r_t})$ is a monotonically decreasing function of $t$, the generalization error decreases monotonically in the first phase of learning. At the end of first phase of learning $\phi(t_1) = O(\frac{1}{n})$, therefore the generalization error is $\mathcal{E}(\alpha_{t_1}) = \mathcal{E}(\alpha_\infty) + O(\frac{1}{n})$. As a summary of these statements we have the following corollary.

**Corollary 3.2** *In the first phase of learning, the complexity error is dominated by the approximation error, and within an order of $O(\frac{1}{n})$, the generalization error decreases monotonically in the learning process to $\mathcal{E}(\alpha_\infty) + O(\frac{1}{n})$ at the end of first phase.*     □

For $t > t_1$, we can show by Theorem 3.1 that the generalization dynamics is given by the following equation, where $\delta_{t_1} \equiv \alpha(t_1) - \alpha^*$,

$$\mathcal{E}_G(\alpha_{t_1+t}) = \mathcal{E}_G(\alpha_0) - \frac{2\sigma^2}{n}\sum_{i=1}^{d}(1 - \epsilon\lambda_i)^t \left[ 1 - \frac{1}{2}\left(1 + \rho_i^2\right)\left(1 - \epsilon\lambda_i\right)^t \right] + O(n^{-\frac{3}{2}}),$$

where $\rho_i^2 \equiv \lambda_i \delta_i^2(t_1)n/\sigma^2$, which is, with probability approaching one, of order $O(n^0)$.

Without causing confusion, we still use $\phi(\cdot)$ for the new time-varying part of the generalization error. The function $\phi(\cdot)$ has much more complex behavior after $t_1$ than in the first phase of learning. As we will see, it decreases for some time, and finally begins to increase again. In particular, we found the best generalization at that $t$ where $\phi(t)$ is minimized. (It is noted that $\delta_{t_1}$ is a random variable now, and the following statements of the generalization dynamics are in the sense of with probability approaching one as $n \to \infty$.)

Define the optimal stopping time: $t_{\min} \equiv \arg\min\{\mathcal{E}(\alpha_t) : t \in [0,\infty]\}$, i.e., the epoch corresponding to the smallest generalization error. Then we can prove the following corollaries:

**Corollary 3.3** *The optimal stopping time $t_{\min} = O(\ln n)$, provided $\sigma^2 > 0$. In particular, the following inequalities hold:*

1. *$t_\ell \leq t_{\min} \leq t_u$; where $t_\ell \equiv t_1 + \min_i \frac{\ln(1+\rho_i^2)}{\ln(1/[1-\epsilon\lambda_i])}$ and $t_u \equiv t_1 + \max_i \frac{\ln(1+\rho_i^2)}{\ln(1/[1-\epsilon\lambda_i])}$ are both finite real numbers. That is, the smallest generalization occurs before the global minimum of the empirical error is reached.*

2. $\phi(\cdot)$ *(tracking the generalization error) decreases monotonically for $t < t_\ell$ and increases monotonically to zero for $t > t_u$; furthermore, $t_{\min}$ is unique if $t_\ell + \frac{\ln 2}{\ln(1/[1-\epsilon\lambda_1])} \geq t_u$.*

3. $-\frac{\sigma^2}{n}\sum_{i=1}^{d}\frac{1}{1+\rho_i^2} \leq \phi(t_{\min}) \leq -\frac{\sigma^2}{n}\frac{2d}{1+\gamma}[\frac{2\gamma}{\gamma+1}\frac{d}{d+\rho^2}]^\gamma$, *where* $\gamma = \frac{\ln(1-\epsilon\lambda_1)}{\ln(1-\epsilon\lambda_d)}$, *and* $\rho^2 = \sum_{i=1}^{d}\rho_i^2$.

In accordance with our earlier definitions, we call the learning process during the time interval between $t_1$ and $t_u$ the second phase of learning; and the rest of time the third phase of learning.

According to Corollary 3.3, for $t > t_u$ sufficiently large, the generalization error is uniformly better than at the global minimum, $\hat{\alpha}$, of the empirical error, although minimum generalization error is achieved between $t_\ell$ and $t_u$. The generalization error is reduced by at least $-\frac{\sigma^2}{n}\frac{2d}{1+\gamma}[\frac{2\gamma}{\gamma+1}\frac{d}{d+\rho^2}]^\gamma$ over that for $\hat{\alpha}$ if we stop training at a proper time. For a fixed number of learning examples, the larger is the ratio $d/n$, the larger is the improvement in generalization error if the algorithm is stopped before the global minimum $\alpha^*$ is reached.

## 4   THE EFFECTIVE SIZE OF THE MACHINE

Our concentration on dynamics and our seeming disregard for complexity do not conflict with the learning-theoretic focus on VC-dimension; in fact, the two attitudes fit nicely together. This section explains the generalization dynamics by introducing the the concept of *effective* complexity of the machine. It is argued that early stopping in effect sets the effective size of the network to a value smaller than its VC-dimension.

The *effective size* of the machine at time $t$ is defined to be $d(t) \equiv \sum_{i=1}^{d}[1-(1-\epsilon\lambda_i)^t]^2$, which increases monotonically to $d$, the VC-dimension of the network, as $t \to \infty$. This definition is justified after the following theorem:

**Theorem 4.1** *Under the assumptions of Theorem 3.1, the following equation holds uniformly for all $\alpha_0$ such that $|\delta| \leq M_n$,*

$$\mathcal{E}(\alpha_t) = \mathcal{E}(\alpha^*, t) + \frac{\sigma^2}{n}d(t) + O(\frac{M_n^2}{n} + n^{-\frac{3}{2}}) \tag{7}$$

*where $\mathcal{E}(\alpha^*, t) = \mathcal{E}(\alpha^*) + \sum_{i=1}^{d}\delta_i^2\lambda_i(1-\epsilon\lambda_i)^{2t}$.*   □

In the limit of learning, we have by letting $t \to \infty$ in the above equation,

$$\mathcal{E}(\hat{\alpha}) = \mathcal{E}(\alpha^*) + \frac{\sigma^2}{n}d + O(n^{-\frac{3}{2}}) \tag{8}$$

Hence, to an order of $O(n^{-1.5})$, the generalization error at the limit of training breaks into two parts: the approximation error $\mathcal{E}(\alpha^*)$, and the complexity error $\frac{d}{n}\sigma^2$. Clearly, the latter is proportional to $d$, the VC-dimension of the network. For all $d$'s larger than necessary, $\mathcal{E}(\alpha^*)$ remains a constant, and the generalization error is determined solely by $\frac{d}{n}$. The term $\mathcal{E}(\alpha^*, t)$ differs from $\mathcal{E}(\alpha^*)$ only in terms of initial error, and is identified to be the approximation error at $t$. Comparison of the above two equations thus shows that it is reasonable to define $\frac{\sigma^2}{n}d(t)$ as the complexity error at $t$, and justifies the definition of $d(t)$ as the effective size of the machine at the same time. The quantity $d(t)$ captures the notion of the degree to which the capacity of the machine is used at $t$. It depends on the machine parameters, the algorithm being used, and the marginal distribution of X. Thus, we see from (7) that the generalization error at epoch $t$ falls into the same two parts as it does at the limit: the approximation error (fitting error) and the complexity error (determined by the effective size of the machine).

As we have show in the last section, during the first phase of learning, the complexity error is of higher order in $n$ compared to the fitting error during the first phase of learning, if the initial error is of order $O(n^0)$ or larger. Thus decrease of the fitting error (which is proportional to the training error, as we will see in the next section) implies the decrease of the generalization error. However,

when the fitting error is brought down to the order $O(\frac{1}{n})$, the decrease of fitting error will no longer imply the decrease of the generalization error. In fact, by the above theorem, the generalization error at $t + t_1$ can be written as

$$\mathcal{E}(\alpha_{t+t_1}) = \mathcal{E}(\alpha^*) + \sum_{i=1}^{d} \lambda_i \delta_i(t_1)^2 (1 - \epsilon\lambda_i)^{2t} + \frac{\sigma^2}{n}d(t) + O(n^{-\frac{3}{2}}).$$

The fitting error and the complexity error compete at order $O(\frac{1}{n})$ during the second phase of learning. After the second the phase of learning, the complexity error dominates the fitting error, still at the order of $O(\frac{1}{n})$. Furthermore, if we define $\kappa \equiv \frac{1}{1+\gamma}[\frac{2\gamma}{\gamma+1}\frac{d}{d+\rho^2}]^\gamma$, then by the above equation and (3.3), we have

**Corollary 4.2** *At the optimal stopping time, the following upper bound on the generalization error holds,*

$$\mathcal{E}(\alpha_{t_{\min}}) \leq \mathcal{E}(\alpha^*) + \frac{\sigma^2}{n}(1 - \kappa)d + O(n^{-\frac{3}{2}}).$$

Since $\kappa$ is a quantity of order $O(n^0)$, $(1 - \kappa)d$ is strictly smaller than $d$. Thus stopping training at $t_{\min}$ has the same effect as using a smaller machine of size less than $(1 - \kappa)d$ and carrying training out to the limit! A more detailed analysis reveals how the effective size of the machine is affected by each neuron in the learning process (omitted due to the space limit).

REMARK:    The concept of effective size of the machine can be defined similarly for an arbitrary starting point. However, to compare the degree to which the capacity of the machine has been used at $t$, one must specify at what distance between the hypothesis $\alpha$ and the truth $\alpha^*$ is such comparison started. While each point in the $d$-dimensional Euclidean space can be regarded as a hypothesis (machine) about $\alpha^*$, it is intuitively clear that each of these machines has a *different* capacity to approximate it. But it is reasonable to think that all of the machines that are on the same sphere $\{\alpha : |\alpha - \alpha^*| = r\}$, for each $r > 0$, have the *same* capacity in approximating $\alpha^*$. Thus, to compare the capacity being used at $t$, we must specify a specific sphere as the starting point; defining the effective size of the machine at $t$ without specifying the starting sphere is clearly meaningless. As we have seen, $r \approx \frac{1}{\sqrt{n}}$ is found to be a good choice for our purposes.

## 5  NETWORK SIZE SELECTION

The next theorem relates the generalization error and training error at each epoch of learning, and forms the basis for choosing the optimal stopping time as well as the best size of the machine during the learning process. In the limit of the learning process, the criterion reduces to the well-known Akaike Information Criterion (AIC) for statistical model selection. Comparison of the two criteria reveals that our criterion will result in better generalization than AIC, since it incorporates the information of each individual neuron rather than just the total number of neurons as in the AIC.

**Theorem 5.1** *Assuming the learning algorithm converges, and the conditions of Theorem 3.1 are satisfied; then the following equation holds:*

$$\mathcal{E}(\alpha_t) = (1 + o(1))\mathbf{E}\,\mathcal{E}_n(\alpha_t) + c(d, t) + o(\tfrac{1}{n}) \tag{9}$$

*where $c(d, t) = \frac{2\sigma^2}{n}\sum_{i=1}^{d}[1 - (1 - \epsilon\lambda_i)^t]$* □

According to this theorem, we find an asymptotically unbiased estimate of $\mathcal{E}(\alpha_t)$ to be $\mathcal{E}_n(\alpha_t) + C(d, t)$ when $\sigma^2$ is known. This results in the following criterion for finding the optimal stopping time and network size:

$$\min\{\mathcal{E}_n(\alpha_t) + C(d, t) : d, t = 1, 2, \ldots\} \tag{10}$$

When $t$ goes to infinity, the above criterion becomes:

$$\min\{\mathcal{E}_n(\hat{\alpha}) + \frac{2\sigma^2 d}{n} : d = 1, 2, \ldots\} \tag{11}$$

which is the AIC for choosing the best size of networks. Therefore, (10) can be viewed as an extension of the AIC to the learning process.     To understand the differences, consider the case when $\xi$ has standard normal distribution $N(0, \sigma^2)$. Under this assumption, the Maximum Likelihood (ML) estimation of the weight vectors is the same as the Mean Square estimation. The AIC was obtained by minimizing $\mathbf{E} \frac{log f_{\hat{\alpha}}(X)}{log f_{\alpha}(X)}$, the Kullback-Leibler distance of the density function $f_{\alpha_{ML}}(X)$ with $\alpha_{ML}$ being the ML estimation of $\alpha$ and that of the true density $f_{\alpha}$. This is equivalent to minimizing $\lim_{t\to\infty} \mathbf{E}\,(Y - f_{\alpha_t}(X))^2 = \mathbf{E}\,(Y - f_{\alpha_{ML}}(X))^2$ (assuming the limit and the expectation are interchangeable). Now it is clear that while AIC chooses networks only at the limit of learning, (10) does this in the whole learning process. Observe that the matrix $\Phi$ is now exactly the Fisher Information Matrix of the density function $f_o(X)$, and $\lambda_i$ is a measure of the capacity of $\psi_i$ in fitting the relation between $X$ and $Y$. Therefore our criterion incorporates the information about each specific neuron provided by the Fisher Information Matrix, which is a measure of how well the data fit the model. This implies that there are two aspects in finding the trade-off between the model complexity and the empirical error in order to minimize the generalization error: one is to have the smallest number of neurons and the other is to minimize the utilization of each neuron. The AIC (and in fact most statistical model selection criteria) are aimed at the former, while our criterion incorporates the two aspects at the same time. We have seen in the earlier discussions that for a given number of neurons, this is done by using the capacity of each neuron in fitting the data only to the degree $1 - (1 - \epsilon\lambda_i)^{t_{\min}}$ rather than to its limit.

## 6   CONCLUDING REMARKS

To the best of our knowledge, the results described in this paper provide for the first time a precise language to describe overtraining phenomena in learning machines such as neural networks. We have studied formally the generalization process of a linear machine when it is trained with a gradient descent algorithm. The concept of *effective size* of a machine was introduced to break the generalization error into two parts: the approximation error and the error caused by a complexity term which is proportional to effective size; the former decreases monotonically and the later increases monotonically in the learning process. When the machine is trained on a finite number of examples, there are in general three distinct phases of learning according to the relative magnitude of the fitting and complexity errors. In particular, there exists an optimal stopping time $t_{\min} = O(\ln n)$ for minimizing generalization error which occurs before the global minimum of the empirical error is reached. These results lead to a generalization of the AIC in which the effect of certain network parameters and time of learning are together taken into account in the network size selection process.

For practical application of neural networks, these results demonstrate that training a network to its limits is not desirable. From the learning-theoretic point of view, the concept of effective dimension of a network tells us that we need more than the VC-dimension of a machine to describe the generalization properties of a machine, except in the limit of learning.

The generalization of the AIC reveals some unknown facts in statistical model selection theory: namely, the generalization error of a network is affected not only by the number of parameters but also by the degree to which each parameter is actually used in the learning process. Occam's principle therefore stands in a subtler form: Make minimal *use* of the capacity of a network for encoding the information provided by learning samples.

Our results hold for weaker assumptions than were made herein about the distributions of $X$ and $\xi$. The case of machines that have vector (rather than scalar) outputs is a simple generalization. Also, our theorems have recently been generalized to the case of general nonlinear machines and are not restricted to the squared error loss function.

While the problem of inferring a rule from the observational data has been studied for a long time in learning theory as well as in other context such as in Linear and Nonlinear Regression, the

study of the problem as a dynamical process seems to open a new avenue for looking at the problem. Many problems are open. For example, it is interesting to know what could be learned from a finite number of examples in a finite number of iterations in the case where the size of the machine is not small compared to the sample size.

**Acknowledgments**

C. Wang thanks Siemens Corporate Research for support during the summer of 1992 when this research was initiated. The work of C. Wang and S. Venkatesh has been supported in part by the Air Force Office of Scientific Research under grant F49620-93-1-0120.

# References

[1] Akaike, H. (1974) Information theory and an extension of the maximum likelihood principle. *Second International Symposium on Information Theory*, Ed. B.N. Krishnaiah, North Holland, Amsterdam, 27-41.

[2] Baldi, P. and Y. Chauvin (1991) Temporal evolution of generalization during learning in linear networks. *Neural Communication.* 3,589-603.

[3] Chow, G. C. (1981) A comparison of the information and posterior probability criteria for model selection. *Journal of Econometrics* 16, 21-34.

[4] Hansen, Lars Kai (1993) Stochastic linear learning: exact test and training error averages. *Neural Networks*, 4, 393-396.

[5] Haussler, D. (1989) Decision theoretical generalization of the PAC model for neural networks and other learning applications. Preprint.

[6] Heskes, Tom M. and Bert Kappen (1991) Learning processes in neural networks. *Physical Review A*, Vol 44, No. 4, 2718- 2726.

[7] Kroght, Anders and John A. Herts Generalization in a linear perceptron in the presence of noise. Preprint.

[8] Nilsson, N. J. *Learning Machines.* New York: McGraw Hill.

[9] Pinelis, I., and S. Utev (1984) Estimates of moments of sums of independent random variables. *Theory of Probability and Its Applications.* 29 (1984) 574-577.

[10] Rissanen, J. (1987) Stochastic complexity. *J. Royal Statistical Society.* Series B, Vol. 49, No. 3, 223-265.

[11] Schwartz, G. (1978) Estimating the dimension of a model. *Annals of Statistics* 6, 461-464.

[12] Sazonov, V. (1982). On the accuracy of normal approximation. *Journal of multivariate analysis.* 12, 371-384.

[13] Senatov, V. (1980) Uniform estimates of the rate of convergence in the multi-dimensional central limit theorem. *Theory of Probability and Its Applications.* 25 (1980) 745-758.

[14] Vapnik, V. (1992) Measuring the capacity of learning machines (I). Preprint.

[15] Weigend, S.A. and Rumelhart (1991). Generalization through minimal networks with application to forcasting. *INTERFACE'91-23rd Symposium on the Interface: Computing Science and Statistics*, ed. E. M., Keramidas, pp362-370. Interface Foundation of North America.
